# Use of Bad Training Data For Better Predictions

**Tal Grossman**
Complex Systems Group (T13) and CNLS
LANL, MS B213 Los Alamos N.M. 87545
Alan Lapedes
Complex Systems Group (T13)
LANL, MS B213 Los Alamos N.M. 87545
and The Santa Fe Institute, Santa Fe, New Mexico

## Abstract

We show how randomly scrambling the output classes of various fractions of the training data may be used to improve predictive accuracy of a classification algorithm. We present a method for calculating the "noise sensitivity signature" of a learning algorithm which is based on scrambling the output classes. This signature can be used to indicate a good match between the complexity of the classifier and the complexity of the data. Use of noise sensitivity signatures is distinctly different from other schemes to avoid over-training, such as cross-validation, which uses only part of the training data, or various penalty functions, which are not data-adaptive. Noise sensitivity signature methods use all of the training data and are manifestly data-adaptive and non-parametric. They are well suited for situations with limited training data.

## 1 INTRODUCTION

A major problem of pattern recognition and classification algorithms that learn from a training set of examples is to select the complexity of the model to be trained. How is it possible to avoid an overparameterized algorithm from "memorizing" the training data? The dangers inherent in over-parameterization are typically

illustrated by analogy to the simple numerical problem of fitting a curve to data points drawn from a simple function. If the fit is with a high degree polynomial then prediction on new points, i.e. generalization, can be quite bad, although the training set accuracy is quite good. The wild oscillations in the fitted function, needed to acheive high training set accuracy, cause poor predictions for new data. When using neural networks, this problem has two basic aspects. One is how to choose the optimal architecture (e.g. the number of layers and units in a feed forward net), the other is to know when to stop training. Of course, these two aspects are related: Training a large net to the highest training set accuracy usually causes overfitting. However, when training is stopped at the "correct" point (where train-set accuracy is lower), large nets are generalizing as good as, or even better than, small networks (as observed e.g. in Weigend 1994). This prompts serious consideration of methods to avoid overparameterization. Various methods to select network architecture or to decide when to stop training have been suggested. These include: (1) use of a penalty function (*c.f.* Weigend et al. 1991). (2) use of cross validation (Stone 1974). (3) minimum description length methods (Rissanen 1989), or (4) "pruning" methods (e.g. Le Cun et al. 1990).

Although all these methods are effective to various degrees, they all also suffer some form of non-optimality:

(1) various forms of penalty function have been proposed and results differ between them. Typically, using a penalty function is generally preferable to not using one. However, it is not at all clear that there exists one "correct" penalty function and hence any given penalty function is usually not optimal. (2) Cross validation holds back part of the training data as a separate valdiation set. It therefore works best in the situation where use of smaller training sets, and use of relatively small validation sets, still allows close approximation to the optimal classifier. This is not likely to be the case in a significantly data-limited regime. (3) MDL methods may be viewed as a form of penalty function and are subject to the issues in point (1) above. (4) pruning methods require training a large net, which can be time consuming, and then "de-tuning" the large network using penalty functions. The issues expressed in point(1) above apply.

We present a new method to avoid overfitting that uses "noisy" training data where some of the output classes for a fraction of the data are scrambled. We describe how to obtain the "noise sensitivity signature" of a classifier (with its learning algorithm), which is based on the scrambled data. This new methodology is not computationally cheap, but neither is it prohibitively expensive. It can provide an alternative to methods (1)-(4) above that (i) can test any complexity parameter of any classifying algorithm (i.e. the architecture, the stopping criterion etc.) (ii) uses all the training data, and (iii) is data adaptive, in contrast to fixed penalty/pruning functions.

## 2   A DETAILED DESCRIPTION OF THE METHOD

Define a "Learning Algorithm" $L(S, P)$, as any procedure which produces a classifier $f(x)$, which is a (discrete) function over a given input space $X$ ($x \in X$). The input of the learning algorithm $L$ is a *Training Set S* and a set of parameters $P$. The training set $S$ is a set of $M$ examples, each example is a pair of an input instance $x_i$

and the desired output $y_i$ associated with it $(i = 1..M)$. We assume that the desired output represents an unknown "target function" $f^*$ which we try to approximate, i.e. $y_i = f^*(x_i)$. The set of parameters $P$ includes all the relevant parameters of the specific learning algorithm and architecture used. When using a feed-forward neural network classifier this set usually includes the size of the network, its connectivity pattern, the distribution of the initial weights and the learning parameters (e.g. the learning rate and momentum term size in usual back-propagation). Some of these parameters determine the "complexity" of the classifiers produced by the learning algorithm, or the set of functions $f$ that are realizable by $L$. The number of hidden units in a two layer perceptron, for example, determines the number of free parameters of the model (the weights) that the learning algorithm will fit to the data (the training set). In general, the output of $L$ can be any classifier: a neural network, a decision tree, boolean formula etc. The classifier $f$ can also depend on some random choices, like the initial choice of weights in many network learning algortihm. It can also depend, like in pruning algorithms on any "stopping criterion" which may also influence its complexity.

## 2.1  PRODUCING $f_p^\mu$

The classification task is given as the training set $S$. The first step of our method is to prepare a set of noisy, or partially scrambled realizations of $S$. We define $S_p^\mu$ as one particular such realization, in which for fraction $p$ of the $M$ examples the desired *output* values (classes) are changed. In this work we consider only binary classification tasks, which means that we choose $pM$ examples at random for which $y_i^\mu = 1 - y_i$. For each noise level $p$ and set of $n$ such realizations $S_p^\mu$ $(\mu = 1..n)$ is prepared, each with a different random choice of scrambled examples. Practically, 8-10 noise levels in the range $p = 0.0 - 0.4$, with $n \sim 4 - 10$ realizations of $S_p^\mu$ for each level are enough. The second step is to apply the learning algorithm to each of the different $S_p^\mu$ to produce the corresponding classifiers, which are the boolean functions $f_p^\mu = L(S_p^\mu, P)$.

## 2.2  NOISE SENSITIVITY MEASURES

Using the set of $f_p^\mu$, three quantities are measured for each noise level $p$:

- The average performance on the original (noise free) training set $S$. We define the average noise-free error as

$$E_f(p) = \frac{1}{Mn} \sum_{\mu}^{n} \sum_{i}^{M} |f_p^\mu(x_i) - y_i| \tag{1}$$

   And the noise-free pereformance, or score as $Q_f(p) = 1 - E_f(p)$.

- In a similar way, we define the average error on the noisy training-sets $S_p^\mu$:

$$E_n(p) = \frac{1}{Mn} \sum_{\mu}^{n} \sum_{i}^{M} |f_p^\mu(x_i) - y_i^\mu| \tag{2}$$

   Note that the error of each classifier $f_p^\mu$ is measured on the training set by which it was created. The noisy-set performance is then defined as $Q_n(p) = 1 - E_n(p)$.

- The average functional distance between classifiers. The functional distance between two classifiers, or boolean functions, $d(f, g)$ is the probability of $f(x) \neq g(x)$. For a uniform input distribution, it is simply the fraction of the input space $X$ for which $f(x) \neq g(x)$. In order to approximate this quantity, we can use another set of examples. In contrast with validation set methods, these examples need not be classified, i.e. we only need a set of inputs $x$, without the target outputs $y$, so we can usually use an "artificial" set of $m$ random inputs. Although, in principle at least, these $x$ instances should be taken from the same distribution as the original task examples. The approximated distance between two classifiers is therefore

$$d(f, g) = \frac{1}{m} \sum_{i}^{m} |f(x_i) - g(x_i)| \tag{3}$$

We then calculate the average distance, $D(p)$, between the $n$ classifiers $f_p^\mu$ obtained for each noise level $p$:

$$D(p) = \frac{2}{n(n-1)} \sum_{\mu > \nu}^{n} d(f_p^\mu, f_p^\nu) \tag{4}$$

## 3    NOISE SENSITIVITY BEHAVIOR

Observing the three quantities $Q_f(p)$, $Q_n(p)$ and $D(p)$, can we distinguish between an overparametrized classifier and a "well tuned" one ? Can we use this data in order to choose the best generalizer out of several candidates ? Or to find the right point to stop the learning algorithm $L$ in order to achieve better generalization ?   Lets estimate how the plots of $Q_f, Q_n$ and $D$ vs. $p$, which we call the "Noise Sensitivity Signature" (NSS) of the algorithm $L$, look like in several different scenarios.

### 3.1    $D(p)$

The average functional distance between realizations, $D(p)$, measures the sensitivity of the classifier (or the model) to noise. An over-parametrized architecture is expected to be very sensitive to noise since it is capable of changing its classification boundary to learn the scrambled examples. Different realizations of the noisy training set will therefore result in different classifiers.

On the other hand, an under-parametrized classifier should be stable against at least a small amount of noise. Its classification boundary will not change when a few examples change their class. Note, however, that if the training set is not very "dense", an under-parametrized architecture can still yield different classifiers, even when trained on a noise free training set (e.g. when using BP with different initial weights). Therefore, it may be possible to observe some "background variance", i.e. non-zero average distance for small (down to zero) noise levels for under-parametrized classifiers.

### 3.2    $Q_f(p)$ AND $Q_n(p)$

Similar considerations apply for the two quantities $Q_f(p)$ and $Q_n(p)$. When the training set is large enough, an under-parametrized classifier cannot "follow" all

the changed examples. Therefore most of them just add to the training error. Nevertheless, its performance on the noise free training set, $Q_f(p)$, will not change much. As a result, when increasing the noise level $p$ from zero (where $Q_f(p) = Q_n(p)$), we should find $Q_f(p) > Q_n(p)$ up to a high noise level - where the decision boundary has changed enough so the error on the original training set becomes larger than the error on the actual noisy set. The more parameters our model has, the sooner (i.e. smaller $p$) it will switch to the $Q_f(p) < Q_n(p)$ state. If a network starts with $Q_f(p) = Q_n(p)$ and then exhibits a behavior with $Q_f(p) < Q_n(p)$, this is a signature of overparameterization.

## 3.3    THE TRAINING SET

In addition to the set of parameters $P$ and the learning algorithm itself, there is another important factor in the learning process. This is the training set $S$. The dependence on $M$, the number of examples is evident. When $M$ is not large enough, the training set does not provide enough data in order to capture the full complexity of the original task. In other words, there are not enough constraints - to approximate well the target function $f^\star$. Therefore overfitting will occur for smaller classifier complexity and the optimal network will be smaller.

## 4    EXPERIMENTAL RESULTS

To demonstrate the possible outcomes of the method described above in several cases, we have performed the following experiment. A random neural network "teacher" was created as the target function $f^\star$. This is a two layer perceptron with 20 inputs, 5 hidden units and one output. A set of $M$ random binary input examples was created and the teacher network was used to classify the training examples. Namely, a desired output $y_i$ was obtained by recording the output of the teacher net when input $x_i$ was presented to the network, and the output was calculated by applying the usual feed forward dynamincs:

$$y_j = sgn(\sum w_{jk} x_k) \qquad (5)$$

This binary threshold update rule is applied to each of the network's units $j$, i.e the hidden and the output units. The weights of the teacher were chosen from a uniform distribution [-1,1]. No threshold (bias weights) were used.

The set of scrambled training sets $S_p^\mu$ was produced as explained above and different network architectures were trained on it to produce the set of classifiers $f_p^\mu$. The learning networks are standard two layer networks of sigmoid units, trained by conjugate gradient back-propagation, using a quadratic error function with tolerance, i.e. if the difference between an output of the net and the desired 0 or 1 target is smaller than the tolerance (taken as 0.2 in our experiment) it does not contribute to the error. The tolerance is, of course, another parameter which may influences the complexity of the resulting network, however, in this experiment it is fixed.

The quantities $Q_f(p)$, $Q_n(p)$ and $D(p)$ were calculated for networks with 1,2,3,..7 hidden units (1 hidden unit means just a perceptron, trained with the same error function). In our terminology, the architecture specification is part of the set of

| hidden units | Training Set Size | | |
|---|---|---|---|
| | 400 | 700 | 1024 |
| 1 | 0.81 (0.04) | 0.81 (0.001) | 0.82 (0.001) |
| 2 | 0.81 (0.04) | 0.84 (0.05) | 0.86 (0.04) |
| 3 | 0.78 (0.02) | 0.82 (0.06) | 0.90 (0.03) |
| 4 | 0.77 (0.03) | 0.81 (0.05) | 0.90 (0.03) |
| 5 | 0.74 (0.03) | 0.79 (0.03) | 0.87 (0.04) |
| 6 | 0.74 (0.01) | 0.80 (0.05) | 0.89 (0.03) |
| 7 | 0.71 (0.01) | 0.76 (0.02) | 0.85 (0.05) |

Table 1: The prediction rate for 1..7 hidden units, averaged on 4 nets that were trained on the noisefree training set of size $M = 400, 700, 1024$ (the standard deviation is given in parenthesis).

parameters $P$ that is input to the learning algorithm $L$. The goal is to identify the "correct" architecture according to the behavior of $Q_f, Q_n$ and $D$ with $p$.

The experiment was done with three training set sizes $M = 400$, 700 and 1024. Another set of $m = 1000$ random examples was used to calculate $D$. As an "external control" this set was also classified by the teacher network and was used to measure the generalization (or prediciton rate) of the different learning networks. The prediction rate, for the networks trained on the noise free training set (averaged over 4 networks, trained with different random initial weights) is given for the 1 to 7 hidden unit architectures, for the 3 sizes of $M$, in Table 1. The noise sensitivity signatures of three architectures trained with $M = 400$ (1,2,3 hidden units) and with $M = 1024$ examples (2,4,6 units) are shown in Figure 1. Compare these (representative) results with the expected behaviour of the NSS as described qualitatively in the previous section.

# 5   CONCLUSIONS and DISCUSSION

We have introduced a method of testing a learning model (with its learning algorithm) against a learning task given as a finite set of examples, by producing and characterizing its "noise sensitivity signature". Relying on the experimental results presented here, and similar results obtained with other (less artificial) learning tasks and algorithms, we suggest some guidelines for using the NSS for model tuning:

1. If $D(p)$ approaches zero with $p \to 0$, or if $Q_f(p)$ is significantly better than $Q_n(p)$ for noise levels up to 0.3 or more - the network/model complexity can be safely inreased.

2. If $Q_f(p) < Q_n(p)$ already for small levels of noise (say 0.2 or less) - **reduce** the network complexity.

3. In more delicate situations: a "good" model will have at least a trace of concavity in $D(p)$. A clearly convex $D(p)$ probably indicates an over-parametrized model. In a "good" model choice, $Q_n(p)$ will follow $Q_f(p)$ closely, from below, up to a high noise level.

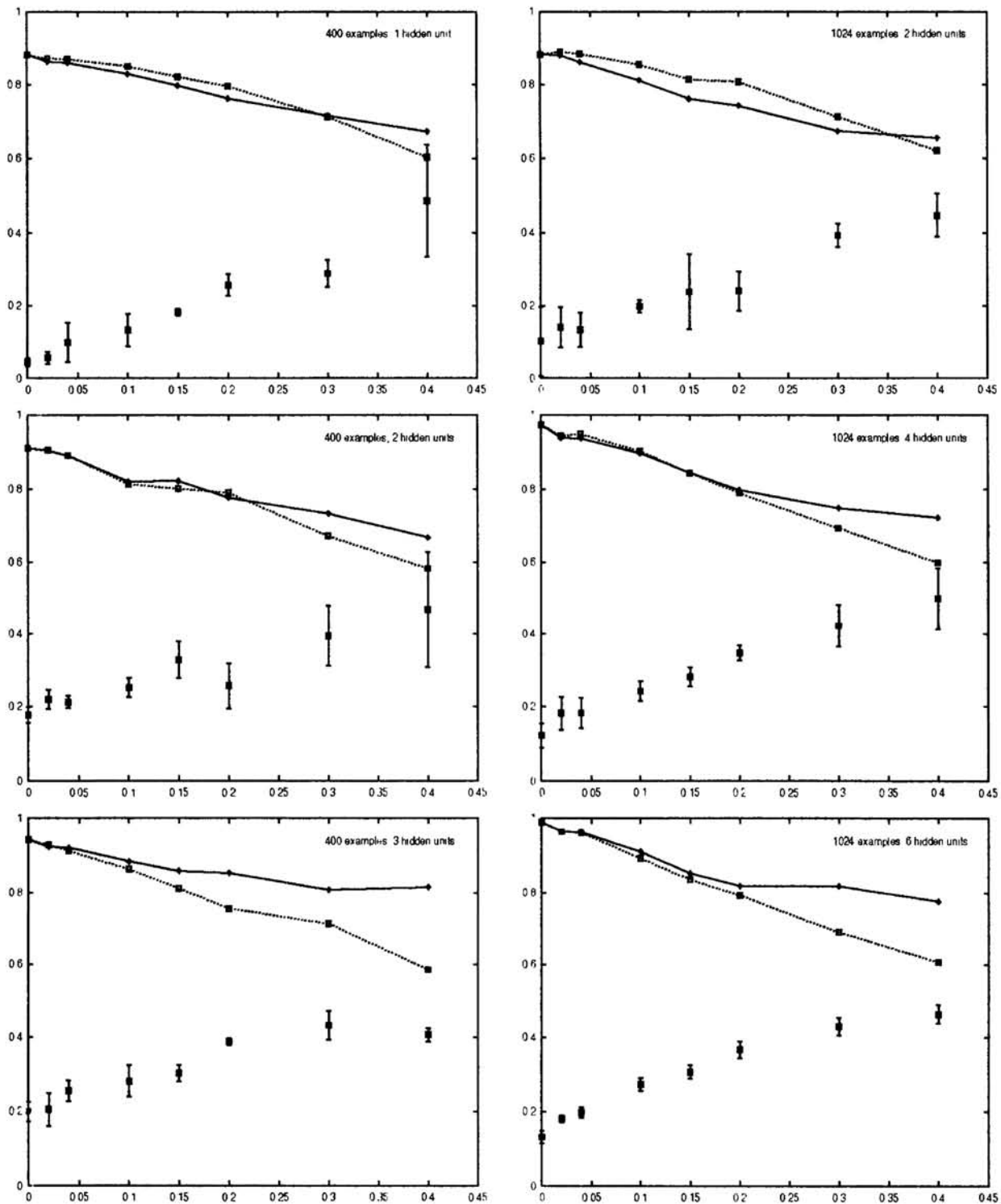

Figure 1: The signatures ($Q$ and $D$ vs. $p$) of networks with 1,2,3 hidden units (top to bottom) trained on M=400 examples (left), and networks with 2,4,6 hidden units trained on M=1024 examples. The (noisy) training set score $Q_n(p)$ is plotted with full line, the noise free score $Q_f(p)$ with dotted line, and the average functional distance $D(p)$ with error bars (representing the standard deviation of the distance).

### 5.1   Advanatages of the Method

1. The method uses all the data for training. Therefore we can extract all the available information. Unlike validation set methods - there is no need to spare part of the examples for testing (note that classified examples are not needed for the functional distance estimation). This may be an important advantage when the data is limited. As the experiment presented here shows: taking 300 examples out of the 1024 given, may result in choosing a smaller network that will give inferior prediction (see table 1). Using "delete-1 cross-validation" will minimize this problem but will need at least as much computation as the NSS calculation in order to achieve reliable prediction estimation.

2. It is an "external" method, i.e. independent of the classifier and the training algorithm. It can be used with neural nets, decision trees, boolean circuits etc. It can evaluate different classifiers, algorithms or stopping/prunning criteria.

### 5.2   Disadvantages

1. Computationally expensive (but not prohibitively so). In principle one can use just a few noise levels to reduce computational cost.

2. Presently requires a subjective decision in order to identify the signature, unlike cross-validation methods which produce one number. In some situations, the noise sensitivity signature gives no clear distinction between similar architectures. In these cases, however, there is almost no difference in their generalization rate.

### Acknowledgements

We thank David Wolpert, Michael Perrone and Jerom Friedman for many iluminating discussions and usefull comments. We also thank Rob Farber for his invaluable help with software and for his assistance with the Connection Machine.

### Referencess

Le Cun Y., Denker J.S. and Solla S. (1990), in Adv. in NIPS 2, Touretzky D.S. ed. (Morgan Kaufmann 1990) 598.

Rissanen J. (1989), *Stochastic Complexity in Statistical Inquiry* (World Scientific 1989).

Stone M. (1974), J.Roy.Statist.Soc.Ser.B **36** (1974) 111.

Wiegend A.S. (1994), in the *Proc. of the 1993 Connectionist Models Summer School*, edited by M.C. Mozer, P. Smolensky, D.S. Touretzky, J.L. Elman and A.S. Weigend, pp. 335-342 (Erlbaum Associates, Hillsdale NJ, 1994).

Wiegend A.S., Rummelhart D. and Huberman B.A. (1991), in Adv. in NIPS 3, Lippmann et al. eds. (Morgen Kaufmann 1991) 875.